# The Laplacian PDF Distance: A Cost Function for Clustering in a Kernel Feature Space

**Robert Jenssen**[1]*, **Deniz Erdogmus**[2], **Jose Principe**[2], **Torbjørn Eltoft**[1]

[1]Department of Physics, University of Tromsø, Norway
[2]Computational NeuroEngineering Laboratory, University of Florida, USA

## Abstract

A new distance measure between probability density functions (pdfs) is introduced, which we refer to as the Laplacian pdf distance. The Laplacian pdf distance exhibits a remarkable connection to Mercer kernel based learning theory via the Parzen window technique for density estimation. In a kernel feature space defined by the eigenspectrum of the Laplacian data matrix, this pdf distance is shown to measure the cosine of the angle between cluster mean vectors. The Laplacian data matrix, and hence its eigenspectrum, can be obtained automatically based on the data at hand, by optimal Parzen window selection. We show that the Laplacian pdf distance has an interesting interpretation as a risk function connected to the probability of error.

## 1 Introduction

In recent years, spectral clustering methods, i.e. data partitioning based on the eigenspectrum of kernel matrices, have received a lot of attention [1, 2]. Some unresolved questions associated with these methods are for example that it is not always clear which cost function that is being optimized and that is not clear how to construct a proper kernel matrix.

In this paper, we introduce a well-defined cost function for spectral clustering. This cost function is derived from a new information theoretic distance measure between cluster pdfs, named the Laplacian pdf distance. The information theoretic/spectral duality is established via the Parzen window methodology for density estimation. The resulting spectral clustering cost function measures the cosine of the angle between cluster mean vectors in a Mercer kernel feature space, where the feature space is determined by the eigenspectrum of the Laplacian matrix. A principled approach to spectral clustering would be to optimize this cost function in the feature space by assigning cluster memberships. Because of space limitations, we leave it to a future paper to present an actual clustering algorithm optimizing this cost function, and focus in this paper on the theoretical properties of the new measure.

An important by-product of the theory presented is that a method for learning the Mercer kernel matrix via optimal Parzen windowing is provided. This means that the Laplacian matrix, its eigenspectrum and hence the feature space mapping can be determined automatically. We illustrate this property by an example.

We also show that the Laplacian pdf distance has an interesting relationship to the probability of error.

In section 2, we briefly review kernel feature space theory. In section 3, we utilize the Parzen window technique for function approximation, in order to introduce the new Laplacian pdf distance and discuss some properties in sections 4 and 5. Section 6 concludes the paper.

## 2    Kernel Feature Spaces

Mercer kernel-based learning algorithms [3] make use of the following idea: via a nonlinear mapping

$$\mathbf{\Phi} : R^d \to \mathcal{F}, \quad \mathbf{x} \to \mathbf{\Phi}(\mathbf{x}) \tag{1}$$

the data $\mathbf{x}_1, \ldots, \mathbf{x}_N \in R^d$ is mapped into a potentially much higher dimensional feature space $\mathcal{F}$. For a given learning problem one now considers the same algorithm in $\mathcal{F}$ instead of in $R^d$, that is, one works with $\mathbf{\Phi}(\mathbf{x}_1), \ldots, \mathbf{\Phi}(\mathbf{x}_N) \in \mathcal{F}$.

Consider a symmetric kernel function $k(\mathbf{x}, \mathbf{y})$. If $k : \mathcal{C} \times \mathcal{C} \to R$ is a continuous kernel of a positive integral operator in a Hilbert space $L_2(\mathcal{C})$ on a compact set $\mathcal{C} \in R^d$, i.e.

$$\forall \psi \in L_2(\mathcal{C}) : \int_{\mathcal{C}} k(\mathbf{x}, \mathbf{y}) \psi(\mathbf{x}) \psi(\mathbf{y}) d\mathbf{x} d\mathbf{y} \geq 0, \tag{2}$$

then there exists a space $\mathcal{F}$ and a mapping $\mathbf{\Phi} : R^d \to \mathcal{F}$, such that by Mercer's theorem [4]

$$k(\mathbf{x}, \mathbf{y}) = \langle \mathbf{\Phi}(\mathbf{x}), \mathbf{\Phi}(\mathbf{y}) \rangle = \sum_{i=1}^{N_{\mathcal{F}}} \lambda_i \phi_i(\mathbf{x}) \phi_i(\mathbf{y}), \tag{3}$$

where $\langle \cdot, \cdot \rangle$ denotes an inner product, the $\phi_i$'s are the orthonormal eigenfunctions of the kernel and $N_{\mathcal{F}} \leq \infty$ [3]. In this case

$$\mathbf{\Phi}(\mathbf{x}) = [\sqrt{\lambda_1} \phi_1(\mathbf{x}), \sqrt{\lambda_2} \phi_2(\mathbf{x}), \ldots]^T, \tag{4}$$

can potentially be realized.

In some cases, it may be desirable to realize this mapping. This issue has been addressed in [5]. Define the $(N \times N)$ Gram matrix, $\mathbf{K}$, also called the affinity, or kernel matrix, with elements $K_{ij} = k(\mathbf{x}_i, \mathbf{x}_j)$, $i, j = 1, \ldots, N$. This matrix can be diagonalized as $\mathbf{E}^T \mathbf{K} \mathbf{E} = \mathbf{\Lambda}$, where the columns of $\mathbf{E}$ contains the eigenvectors of $\mathbf{K}$ and $\mathbf{\Lambda}$ is a diagonal matrix containing the non-negative eigenvalues $\tilde{\lambda}_1, \ldots, \tilde{\lambda}_N$, $\tilde{\lambda}_1 \geq \cdots \geq \tilde{\lambda}_N$. In [5], it was shown that the eigenfunctions and eigenvalues of (4) can be approximated as $\phi_j(\mathbf{x}_i) \approx \sqrt{N} e_{ji}$, $\lambda_j \approx \frac{\tilde{\lambda}_j}{N}$, where $e_{ji}$ denotes the $i$th element of the $j$th eigenvector. Hence, the mapping (4), can be approximated as

$$\mathbf{\Phi}(\mathbf{x}_i) \approx [\sqrt{\tilde{\lambda}_1} e_{1i}, \ldots, \sqrt{\tilde{\lambda}_N} e_{Ni}]^T. \tag{5}$$

Thus, the mapping is based on the eigenspectrum of $\mathbf{K}$. The feature space data set may be represented in matrix form as $\underline{\mathbf{\Phi}}^{N \times N} = [\mathbf{\Phi}(\mathbf{x}_1), \ldots, \mathbf{\Phi}(\mathbf{x}_N)]$. Hence, $\underline{\mathbf{\Phi}} = \mathbf{\Lambda}^{\frac{1}{2}} \mathbf{E}^T$. It may be desirable to truncate the mapping (5) to $C$-dimensions. Thus,

only the $C$ first rows of $\underline{\underline{\Phi}}$ are kept, yielding $\underline{\underline{\hat{\Phi}}}$. It is well-known that $\hat{\mathbf{K}} = \underline{\underline{\hat{\Phi}}}^T \underline{\underline{\hat{\Phi}}}$ is the best rank-$C$ approximation to $\mathbf{K}$ wrt. the Frobenius norm [6].

The most widely used Mercer kernel is the radial-basis-function (RBF)

$$k(\mathbf{x}, \mathbf{y}) = \exp\left\{ -\frac{||\mathbf{x} - \mathbf{y}||^2}{2\sigma^2} \right\}. \tag{6}$$

## 3 Function Approximation using Parzen Windowing

Parzen windowing is a kernel-based density estimation method, where the resulting density estimate is continuous and differentiable provided that the selected kernel is continuous and differentiable [7]. Given a set of iid samples $\{\mathbf{x}_1, \ldots, \mathbf{x}_N\}$ drawn from the true density $f(\mathbf{x})$, the Parzen window estimate for this distribution is [7]

$$\hat{f}(\mathbf{x}) = \frac{1}{N} \sum_{i=1}^{N} W_{\sigma^2}(\mathbf{x}, \mathbf{x}_i), \tag{7}$$

where $W_{\sigma^2}$ is the Parzen window, or kernel, and $\sigma^2$ controls the width of the kernel. The Parzen window must integrate to one, and is typically chosen to be a pdf itself with mean $\mathbf{x}_i$, such as the Gaussian kernel

$$W_{\sigma^2}(\mathbf{x}, \mathbf{x}_i) = \frac{1}{(2\pi\sigma^2)^{\frac{d}{2}}} \exp\left\{ -\frac{||\mathbf{x} - \mathbf{x}_i||^2}{2\sigma^2} \right\}, \tag{8}$$

which we will assume in the rest of this paper. In the conclusion, we briefly discuss the use of other kernels.

Consider a function $h(\mathbf{x}) = v(\mathbf{x})f(\mathbf{x})$, for some function $v(\mathbf{x})$. We propose to estimate $h(\mathbf{x})$ by the following generalized Parzen estimator

$$\hat{h}(\mathbf{x}) = \frac{1}{N} \sum_{i=1}^{N} v(\mathbf{x}_i) W_{\sigma^2}(\mathbf{x}, \mathbf{x}_i). \tag{9}$$

This estimator is asymptotically unbiased, which can be shown as follows

$$E_f\left\{ \frac{1}{N} \sum_{i=1}^{N} v(\mathbf{x}_i) W_{\sigma^2}(\mathbf{x}, \mathbf{x}_i) \right\} = \int v(\mathbf{z}) f(\mathbf{z}) W_{\sigma^2}(\mathbf{x}, \mathbf{z}) d\mathbf{z} = [v(\mathbf{x})f(\mathbf{x})] * W_{\sigma^2}(\mathbf{x}), \tag{10}$$

where $E_f(\cdot)$ denotes expectation with respect to the density $f(\mathbf{x})$. In the limit as $N \to \infty$ and $\sigma(N) \to 0$, we have

$$\lim_{\substack{N \to \infty \\ \sigma(N) \to 0}} [v(\mathbf{x})f(\mathbf{x})] * W_{\sigma^2}(\mathbf{x}) = v(\mathbf{x})f(\mathbf{x}). \tag{11}$$

Of course, if $v(\mathbf{x}) = 1 \; \forall \mathbf{x}$, then (9) is nothing but the traditional Parzen estimator of $h(\mathbf{x}) = f(\mathbf{x})$. The estimator (9) is also asymptotically consistent provided that the kernel width $\sigma(N)$ is annealed at a sufficiently slow rate. The proof will be presented in another paper.

Many approaches have been proposed in order to optimally determine the size of the Parzen window, given a finite sample data set. A simple selection rule was proposed by Silverman [8], using the mean integrated square error (MISE) between the estimated and the actual pdf as the optimality metric:

$$\sigma_{\text{opt}} = \sigma_X \left\{ 4N^{-1}(2d + 1)^{-1} \right\}^{\frac{1}{d+4}}, \tag{12}$$

where $d$ is the dimensionality of the data and $\sigma_X^2 = d^{-1} \sum_i \Sigma_{X_{ii}}$, where $\Sigma_{X_{ii}}$ are the diagonal elements of the sample covariance matrix. More advanced approximations to the MISE solution also exist.

# 4   The Laplacian PDF Distance

Cost functions for clustering are often based on distance measures between pdfs. The goal is to assign memberships to the data patterns with respect to a set of clusters, such that the cost function is optimized.

Assume that a data set consists of two clusters. Associate the probability density function $p(\mathbf{x})$ with one of the clusters, and the density $q(\mathbf{x})$ with the other cluster. Let $f(\mathbf{x})$ be the overall probability density function of the data set. Now define the $f^{-1}$ weighted inner product between $p(\mathbf{x})$ and $q(\mathbf{x})$ as $\langle p, q \rangle_f \equiv \int p(\mathbf{x})q(\mathbf{x})f^{-1}(\mathbf{x})d\mathbf{x}$. In such an inner product space, the Cauchy-Schwarz inequality holds, that is, $\langle p, q \rangle_f^2 \leq \langle p, p \rangle_f \langle q, q \rangle_f$. Based on this discussion, an information theoretic distance measure between the two pdfs can be expressed as

$$D_L = -\log \frac{\langle p, q \rangle_f}{\sqrt{\langle p, p \rangle_f \langle q, q \rangle_f}} \geq 0. \tag{13}$$

We refer to this measure as the Laplacian pdf distance, for reasons that we discuss next. It can be seen that the distance $D_L$ is zero if and only if the two densities are equal. It is non-negative, and increases as the overlap between the two pdfs decreases. However, it does not obey the triangle inequality, and is thus not a distance measure in the strict mathematical sense.

We will now show that the Laplacian pdf distance is also a cost function for clustering in a kernel feature space, using the generalized Parzen estimators discussed in the previous section. Since the logarithm is a monotonic function, we will derive the expression for the argument of the log in (13). This quantity will for simplicity be denoted by the letter "L" in equations.

Assume that we have available the iid data points $\{\mathbf{x}_i\}$, $i = 1, \ldots, N_1$, drawn from $p(\mathbf{x})$, which is the density of cluster $C_1$, and the iid $\{\mathbf{x}_j\}$, $j = 1, \ldots, N_2$, drawn from $q(\mathbf{x})$, the density of $C_2$. Let $h(\mathbf{x}) = f^{-\frac{1}{2}}(\mathbf{x})p(\mathbf{x})$ and $g(\mathbf{x}) = f^{-\frac{1}{2}}(\mathbf{x})q(\mathbf{x})$. Hence, we may write

$$L = \frac{\int h(\mathbf{x})g(\mathbf{x})d\mathbf{x}}{\sqrt{\int h^2(\mathbf{x})d\mathbf{x} \int g^2(\mathbf{x})d\mathbf{x}}}. \tag{14}$$

We estimate $h(\mathbf{x})$ and $g(\mathbf{x})$ by the generalized Parzen kernel estimators, as follows

$$\hat{h}(\mathbf{x}) = \frac{1}{N_1} \sum_{i=1}^{N_1} f^{-\frac{1}{2}}(\mathbf{x}_i)W_{\sigma^2}(\mathbf{x}, \mathbf{x}_i), \quad \hat{g}(\mathbf{x}) = \frac{1}{N_2} \sum_{j=1}^{N_2} f^{-\frac{1}{2}}(\mathbf{x}_j)W_{\sigma^2}(\mathbf{x}, \mathbf{x}_j). \tag{15}$$

The approach taken, is to substitute these estimators into (14), to obtain

$$\begin{aligned}
\int h(\mathbf{x})g(\mathbf{x})d\mathbf{x} &\approx \int \frac{1}{N_1} \sum_{i=1}^{N_1} f^{-\frac{1}{2}}(\mathbf{x}_i)W_{\sigma^2}(\mathbf{x}, \mathbf{x}_i)\frac{1}{N_2} \sum_{j=1}^{N_2} f^{-\frac{1}{2}}(\mathbf{x}_j)W_{\sigma^2}(\mathbf{x}, \mathbf{x}_j) \\
&= \frac{1}{N_1 N_2} \sum_{i,j=1}^{N_1,N_2} f^{-\frac{1}{2}}(\mathbf{x}_i)f^{-\frac{1}{2}}(\mathbf{x}_j) \int W_{\sigma^2}(\mathbf{x}, \mathbf{x}_i)W_{\sigma^2}(\mathbf{x}, \mathbf{x}_j)d\mathbf{x} \\
&= \frac{1}{N_1 N_2} \sum_{i,j=1}^{N_1,N_2} f^{-\frac{1}{2}}(\mathbf{x}_i)f^{-\frac{1}{2}}(\mathbf{x}_j)W_{2\sigma^2}(\mathbf{x}_i, \mathbf{x}_j), \tag{16}
\end{aligned}$$

where in the last step, the convolution theorem for Gaussians has been employed. Similarly, we have

$$\int h^2(\mathbf{x})d\mathbf{x} \approx \frac{1}{N_1^2} \sum_{i,i'=1}^{N_1,N_1} f^{-\frac{1}{2}}(\mathbf{x}_i)f^{-\frac{1}{2}}(\mathbf{x}_{i'})W_{2\sigma^2}(\mathbf{x}_i,\mathbf{x}_{i'}), \tag{17}$$

$$\int g^2(\mathbf{x})d\mathbf{x} \approx \frac{1}{N_2^2} \sum_{j,j'=1}^{N_2,N_2} f^{-\frac{1}{2}}(\mathbf{x}_j)f^{-\frac{1}{2}}(\mathbf{x}_{j'})W_{2\sigma^2}(\mathbf{x}_j,\mathbf{x}_{j'}). \tag{18}$$

Now we define the matrix $\mathbf{K}_f$, such that

$$K_{f_{ij}} = K_f(\mathbf{x}_i,\mathbf{x}_j) = f^{-\frac{1}{2}}(\mathbf{x}_i)f^{-\frac{1}{2}}(\mathbf{x}_j)K(\mathbf{x}_i,\mathbf{x}_j), \tag{19}$$

where $K(\mathbf{x}_i,\mathbf{x}_j) = W_{2\sigma^2}(\mathbf{x}_i,\mathbf{x}_j)$ for $i,j = 1,\ldots,N$ and $N = N_1 + N_2$. As a consequence, (14) can be re-written as follows

$$L = \frac{\sum_{i,j=1}^{N_1,N_2} K_f(\mathbf{x}_i,\mathbf{x}_j)}{\sqrt{\sum_{i,i'=1}^{N_1,N_1} K_f(\mathbf{x}_i,\mathbf{x}_{i'}) \sum_{j,j'=1}^{N_2,N_2} K_f(\mathbf{x}_j,\mathbf{x}_{j'})}} \tag{20}$$

The key point of this paper, is to note that the matrix $\mathbf{K} = K_{ij} = K(\mathbf{x}_i,\mathbf{x}_j)$, $i,j = 1,\ldots,N$, is the data affinity matrix, and that $K(\mathbf{x}_i,\mathbf{x}_j)$ *is a Gaussian RBF kernel function.* Hence, it is also a kernel function that satisfies Mercer's theorem. Since $K(\mathbf{x}_i,\mathbf{x}_j)$ satisfies Mercer's theorem, the following by definition holds [4]. For any set of examples $\{\mathbf{x}_1,\ldots,\mathbf{x}_N\}$ and any set of real numbers $\psi_1,\ldots,\psi_N$

$$\sum_{i=1}^{N}\sum_{j=1}^{N} \psi_i\psi_j K(\mathbf{x}_i,\mathbf{x}_j) \geq 0, \tag{21}$$

in analogy to (3). Moreover, this means that

$$\sum_{i=1}^{N}\sum_{j=1}^{N} \psi_i\psi_j f^{-\frac{1}{2}}(\mathbf{x}_i)f^{-\frac{1}{2}}(\mathbf{x}_j)K(\mathbf{x}_i,\mathbf{x}_j) = \sum_{i=1}^{N}\sum_{j=1}^{N} \psi_i\psi_j K_f(\mathbf{x}_i,\mathbf{x}_j) \geq 0, \tag{22}$$

hence $K_f(\mathbf{x}_i,\mathbf{x}_j)$ *is also a Mercer kernel.*

Now, it is readily observed that the Laplacian pdf distance can be analyzed in terms of inner products in a Mercer kernel-based Hilbert feature space, since $K_f(\mathbf{x}_i,\mathbf{x}_j) = \langle \mathbf{\Phi}_f(\mathbf{x}_i), \mathbf{\Phi}_f(\mathbf{x}_j) \rangle$. Consequently, (20) can be written as follows

$$L = \frac{\sum_{i,j=1}^{N_1,N_2} \langle \mathbf{\Phi}_f(\mathbf{x}_i), \mathbf{\Phi}_f(\mathbf{x}_j) \rangle}{\sqrt{\sum_{i,i'=1}^{N_1,N_1} \langle \mathbf{\Phi}_f(\mathbf{x}_i), \mathbf{\Phi}_f(\mathbf{x}_{i'}) \rangle \sum_{j,j'=1}^{N_2,N_2} \langle \mathbf{\Phi}_f(\mathbf{x}_j), \mathbf{\Phi}_f(\mathbf{x}_{j'}) \rangle}}$$

$$= \frac{\left\langle \frac{1}{N_1}\sum_{i=1}^{N_1} \mathbf{\Phi}_f(\mathbf{x}_i), \frac{1}{N_2}\sum_{j=1}^{N_2} \mathbf{\Phi}_f(\mathbf{x}_j) \right\rangle}{\sqrt{\left\langle \frac{1}{N_1}\sum_{i=1}^{N_1} \mathbf{\Phi}_f(\mathbf{x}_i), \frac{1}{N_1}\sum_{i'=1}^{N_1} \mathbf{\Phi}_f(\mathbf{x}_{i'}) \right\rangle \left\langle \frac{1}{N_2}\sum_{j=1}^{N_2} \mathbf{\Phi}_f(\mathbf{x}_j), \frac{1}{N_2}\sum_{j'=1}^{N_2} \mathbf{\Phi}_f(\mathbf{x}_{j'}) \right\rangle}}$$

$$= \frac{\langle \mathbf{m}_{1_f}, \mathbf{m}_{2_f} \rangle}{\|\mathbf{m}_{1_f}\|\|\mathbf{m}_{2_f}\|} = \cos \angle(\mathbf{m}_{1_f}, \mathbf{m}_{2_f}), \tag{23}$$

where $\mathbf{m}_{i_f} = \frac{1}{N_i}\sum_{l=1}^{N_i} \mathbf{\Phi}_f(\mathbf{x}_l)$, $i = 1, 2$, that is, the sample mean of the $i$th cluster in feature space.

This is a very interesting result. We started out with a distance measure between densities in the input space. By utilizing the Parzen window method, this distance measure turned out to have an equivalent expression as a measure of the distance between two clusters of data points in a Mercer kernel feature space. In the feature space, the distance that is measured is the cosine of the angle between the cluster mean vectors.

The actual mapping of a data point to the kernel feature space is given by the eigendecomposition of $\mathbf{K}_f$, via (5). Let us examine this mapping in more detail. Note that $f^{\frac{1}{2}}(\mathbf{x}_i)$ can be estimated from the data by the traditional Parzen pdf estimator as follows

$$f^{\frac{1}{2}}(\mathbf{x}_i) = \sqrt{\frac{1}{N} \sum_{l=1}^{N} W_{\sigma_f^2}(\mathbf{x}_i, \mathbf{x}_l)} = \sqrt{d_i}. \tag{24}$$

Define the matrix $\mathbf{D} = \mathrm{diag}(d_1, \ldots, d_N)$. Then $\mathbf{K}_f$ can be expressed as

$$\mathbf{K}_f = \mathbf{D}^{-\frac{1}{2}} \mathbf{K} \mathbf{D}^{-\frac{1}{2}}. \tag{25}$$

Quite interestingly, for $\sigma_f^2 = 2\sigma^2$, this is in fact the *Laplacian data matrix.* [1]

The above discussion explicitly connects the Parzen kernel and the Mercer kernel. Moreover, automatic procedures exist in the density estimation literature to optimally determine the Parzen kernel given a data set. Thus, the Mercer kernel is also determined by the same procedure. Therefore, the mapping by the Laplacian matrix to the kernel feature space can also be determined automatically. We regard this as a significant result in the kernel based learning theory.

As an example, consider Fig. 1 (a) which shows a data set consisting of a ring with a dense cluster in the middle. The MISE kernel size is $\sigma_{\mathrm{opt}} = 0.16$, and the Parzen pdf estimate is shown in Fig. 1 (b). The data mapping given by the corresponding Laplacian matrix is shown in Fig. 1 (c) (truncated to two dimensions for visualization purposes). It can be seen that the data is distributed along two lines radially from the origin, indicating that clustering based on the angular measure we have derived makes sense.

The above analysis can easily be extended to any number of pdfs/clusters. In the $C$-cluster case, we define the Laplacian pdf distance as

$$L = \sum_{i=1}^{C-1} \sum_{j \neq i} \frac{\langle p_i, p_j \rangle_f}{C \sqrt{\langle p_i, p_i \rangle_f \langle p_j, p_j \rangle_f}}. \tag{26}$$

In the kernel feature space, (26), corresponds to all cluster mean vectors being pairwise as orthogonal to each other as possible, for all possible unique pairs.

## 4.1 Connection to the Ng et al. [2] algorithm

Recently, Ng et al. [2] proposed to map the input data to a feature space determined by the eigenvectors corresponding to the $C$ largest eigenvalues of the Laplacian matrix. In that space, the data was normalized to unit norm and clustered by the $C$-means algorithm. We have shown that the Laplacian pdf distance provides a

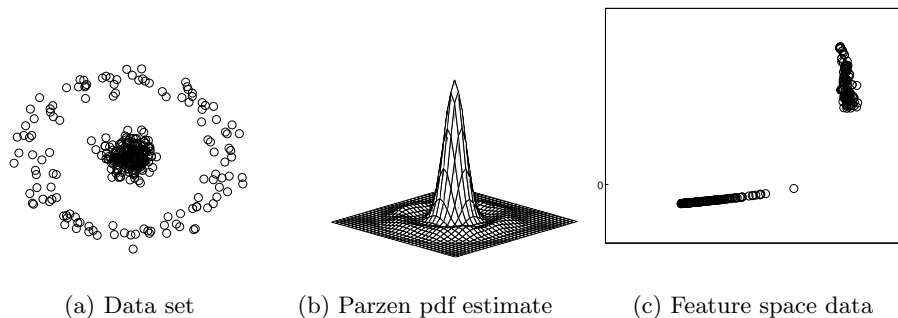

| (a) Data set | (b) Parzen pdf estimate | (c) Feature space data |

Figure 1: The kernel size is automatically determined (MISE), yielding the Parzen estimate (b) with the corresponding feature space mapping (c).

clustering cost function, measuring the cosine of the angle between cluster means, in a related kernel feature space, which in our case can be determined automatically. A more principled approach to clustering than that taken by Ng et al. is to optimize (23) in the feature space, instead of using $C$-means. However, because of the normalization of the data in the feature space, $C$-means can be interpreted as clustering the data based on an angular measure. This may explain some of the success of the Ng et al. algorithm; it achieves more or less the same goal as clustering based on the Laplacian distance would be expected to do. We will investigate this claim in our future work. Note that we in our framework may choose to use only the $C$ largest eigenvalues/eigenvectors in the mapping, as discussed in section 2. Since we incorporate the eigenvalues in the mapping, in contrast to Ng et al., the actual mapping will in general be different in the two cases.

## 5   The Laplacian PDF distance as a risk function

We now give an analysis of the Laplacian pdf distance that may further motivate its use as a clustering cost function. Consider again the two cluster case. The overall data distribution can be expressed as $f(\mathbf{x}) = P_1 p(\mathbf{x}) + P_2 q(\mathbf{x})$, were $P_i$, $i = 1, 2$, are the priors. Assume that the two clusters are well separated, such that for $\mathbf{x}_i \in C_1$, $f(\mathbf{x}_i) \approx P_1 p(\mathbf{x}_i)$, while for $\mathbf{x}_i \in C_2$, $f(\mathbf{x}_i) \approx P_2 q(\mathbf{x}_i)$. Let us examine the numerator of (14) in this case. It can be approximated as $\int \frac{p(\mathbf{x})q(\mathbf{x})}{f(\mathbf{x})} d\mathbf{x}$

$$\approx \int_{C_1} \frac{p(\mathbf{x})q(\mathbf{x})}{f(\mathbf{x})} d\mathbf{x} + \int_{C_2} \frac{p(\mathbf{x})q(\mathbf{x})}{f(\mathbf{x})} d\mathbf{x} \approx \frac{1}{P_1} \int_{C_1} q(\mathbf{x}) d\mathbf{x} + \frac{1}{P_2} \int_{C_2} p(\mathbf{x}) d\mathbf{x}. \quad (27)$$

By performing a similar calculation for the denominator of (14), it can be shown to be approximately equal to $\frac{1}{\sqrt{P_1 P_1}}$. Hence, the Laplacian pdf distance can be written as a risk function, given by

$$L \approx \sqrt{P_1 P_2} \left( \frac{1}{P_1} \int_{C_1} q(\mathbf{x}) d\mathbf{x} + \frac{1}{P_2} \int_{C_2} p(\mathbf{x}) d\mathbf{x} \right). \quad (28)$$

Note that if $P_1 = P_2 = \frac{1}{2}$, then $L = 2P_e$, where $P_e$ is the probability of error when assigning data points to the two clusters, that is

$$P_e = P_1 \int_{C_1} q(\mathbf{x}) d\mathbf{x} + P_2 \int_{C_2} p(\mathbf{x}) d\mathbf{x}. \quad (29)$$

Thus, in this case, minimizing $L$ is equivalent to minimizing $P_e$. However, in the case that $P_1 \neq P_2$, (28) has an even more interesting interpretation. In that situation, it can be seen that the two integrals in the expressions (28) and (29) are weighted exactly oppositely. For example, if $P_1$ is close to one, $L \approx \int_{C_2} p(\mathbf{x}) d\mathbf{x}$, while $P_e \approx \int_{C_1} q(\mathbf{x}) d\mathbf{x}$. Thus, the Laplacian pdf distance emphasizes to cluster the most unlikely data points correctly. In many real world applications, this property may be crucial. For example, in medical applications, the most important points to classify correctly are often the least probable, such as detecting some rare disease in a group of patients.

## 6    Conclusions

We have introduced a new pdf distance measure that we refer to as the Laplacian pdf distance, and we have shown that it is in fact a clustering cost function in a kernel feature space determined by the eigenspectrum of the Laplacian data matrix. In our exposition, the Mercer kernel and the Parzen kernel is equivalent, making it possible to determine the Mercer kernel based on automatic selection procedures for the Parzen kernel. Hence, the Laplacian data matrix and its eigenspectrum can be determined automatically too. We have shown that the new pdf distance has an interesting property as a risk function.

The results we have derived can only be obtained analytically using Gaussian kernels. The same results may be obtained using other Mercer kernels, but it requires an additional approximation wrt. the expectation operator. This discussion is left for future work.

**Acknowledgments**.    This work was partially supported by NSF grant ECS-0300340.

## Footnotes

*Corresponding author. Phone: (+47) 776 46493. Email: robertj@phys.uit.no

[1]It is a bit imprecise to refer to $\mathbf{K}_f$ as the Laplacian matrix, as readers familiar with spectral graph theory may recognize, since the definition of the Laplacian matrix is $\mathbf{L} = \mathbf{I} - \mathbf{K}_f$. However, replacing $\mathbf{K}_f$ by $\mathbf{L}$ does not change the eigenvectors, it only changes the eigenvalues from $\lambda_i$ to $1 - \lambda_i$.

## References

[1] Y. Weiss, "Segmentation Using Eigenvectors: A Unifying View," in *International Conference on Computer Vision*, 1999, pp. 975–982.

[2] A. Y. Ng, M. Jordan, and Y. Weiss, "On Spectral Clustering: Analysis and an Algorithm," in *Advances in Neural Information Processing Systems, 14*, 2001, vol. 2, pp. 849–856.

[3] K. R. Müller, S. Mika, G. Rätsch, K. Tsuda, and B. Schölkopf, "An Introduction to Kernel-Based Learning Algorithms," *IEEE Transactions on Neural Networks*, vol. 12, no. 2, pp. 181–201, 2001.

[4] J. Mercer, "Functions of Positive and Negative Type and their Connection with the Theory of Integral Equations," *Philos. Trans. Roy. Soc. London*, vol. A, pp. 415–446, 1909.

[5] C. Williams and M. Seeger, "Using the Nyström Method to Speed Up Kernel Machines," in *Advances in Neural Information Processing Systems 13*, Vancouver, Canada, USA, 2001, pp. 682–688.

[6] M. Brand and K. Huang, "A Unifying Theorem for Spectral Embedding and Clustering," in *Ninth Int'l Workshop on Artificial Intelligence and Statistics*, Key West, Florida, USA, 2003.

[7] E. Parzen, "On the Estimation of a Probability Density Function and the Mode," *Ann. Math. Stat.*, vol. 32, pp. 1065–1076, 1962.

[8] B. W. Silverman, *Density Estimation for Statistics and Data Analysis*, Chapman and Hall, London, 1986.
